# Temporal Coding using the Response Properties of Spiking Neurons

**Thomas Voegtlin**
INRIA - Campus Scientifique, B.P. 239
F-54506 Vandoeuvre-Les-Nancy Cedex, FRANCE
voegtlin@loria.fr

## Abstract

In biological neurons, the timing of a spike depends on the timing of synaptic currents, in a way that is classically described by the Phase Response Curve. This has implications for temporal coding: an action potential that arrives on a synapse has an implicit meaning, that depends on the position of the postsynaptic neuron on the firing cycle. Here we show that this implicit code can be used to perform computations. Using theta neurons, we derive a spike-timing dependent learning rule from an error criterion. We demonstrate how to train an auto-encoder neural network using this rule.

## 1   Introduction

The temporal coding hypothesis states that information is encoded in the precise timing of action potentials sent by neurons. In order to achieve computations in the time domain, it is thus necessary to have neurons spike at desired times. However, at a more fundamental level, it is also necessary to describe how the timings of action potentials received by a neuron are combined together, in a way that is consistent with the neural code.

So far, the main theory has posited that the shape of post-synaptic potentials (PSPs) is relevant for computations [1, 2, 3]. In these models, the membrane potential at the soma of a neuron is a weighted sum of PSPs arriving from dendrites at different times. The spike time of the neuron is defined as the time when its membrane potential first reaches a firing threshold, and it depends on the precise temporal arrangement of PSPs, thus enabling computations in the time domain. Hence, the nature of the temporal code is closely tied to the shape of PSPs. A consequence is that the length of the rising segment of post-synaptic potentials limits the available coding interval [1, 2].

Here we propose a new theory, based on the non-linear dynamics of integrate-and-fire neurons. This theory takes advantage of the fact that the effect of synaptic currents depends on the internal state of the postsynaptic neuron. For neurons spiking regularly, this dependency is classically described by the Phase Response Curve (PRC) [4]. We use theta neurons, which are mathematically equivalent to quadratic integrate-and-fire neurons [5, 6]. In these neuron models, once the potential has crossed the firing threshold, the neuron is still sensitive to incoming currents, which may change the timing of the next spike.

In the proposed model, computations do not rely on the shape of PSPs, which alleviates the restriction imposed by the length of their rising segment. Therefore, we may use a simplified model of synaptic currents; we model synaptic currents as Diracs, which means that we do not take into account synaptic time constants. Another advantage of our model is that computations do not rely on the delays imposed by inter-neuron transmission; this means that it is not necessary to fine-tune delays in order to learn desired spike times.

## 2 Description of the model

### 2.1 The Theta Neuron

The theta neuron is described by the following differential equation:

$$\frac{d\theta}{dt} = (1 - cos\theta) + \alpha I(1 + cos\theta) \tag{1}$$

where $\theta$ is the "potential" of the neuron, and $I$ is a variable input current, measured in radians per unit of time. For convenience, we call units of time 'milliseconds'. The neuron is said to fire everytime $\theta$ crosses $\pi$. The dynamics of the model can be represented on a phase circle (Figure 1). The effect of an input current is not uniform across the circle; currents that occur late (for $\theta$ close to $\pi$) have little effect on $\theta$, while currents that arrive when $\theta$ is close to zero have a much greater effect.

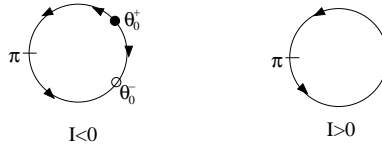

Figure 1: **Phase circle of the theta model**. The neuron fires everytime $\theta$ crosses $\pi$. For $I < 0$ there are two fixed points: An unstable point $\theta_0^+ = arccos\frac{1+\alpha I}{1-\alpha I}$, and an attractor $\theta_0^- = -\theta_0^+$.

### 2.2 Synaptic interactions

The input current $I$ is the sum of a constant current $I_0$ and transient synaptic currents $I_i(t)$, where $i \in 1..N$ indexes the synapses:

$$I = I_0 + \sum_{i=1}^{N} I_i(t) \tag{2}$$

Synaptic currents are modeled as Diracs : $I_i(t) = w_i\delta(t - t_i)$, where $t_i$ is the firing time of presynaptic neuron $i$, and $w_i$ is the weight of the synapse. Transmission delays are not taken into account.

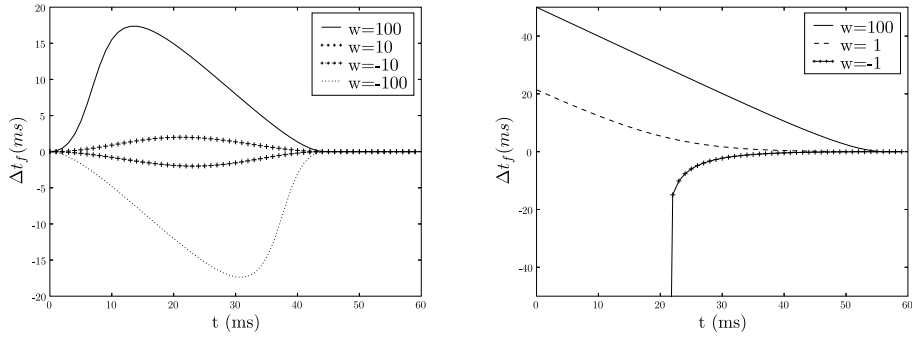

Figure 2: **Response properties of the theta model.** Curves shows the change of firing time $t_f$ of a neuron receiving a Dirac current of weight $w$ at time $t$. Left: For $I_0 > 0$, the neuron spikes regularly ($I_0 = 0.005$, $\theta(0) = -\pi$). If $w$ is small, the curves corresponding to $w > 0$ and $w < 0$ are symmetric; the positive curve is called the Phase Response Curve (PRC). If $w$ is large, curves are no longer symmetric; the portions correspond to the ascending (resp. descending) phase of $sin\,\theta$ have different slopes. Right: Response for $I_0 < 0$. The initial condition is slightly above the unstable equilibrium point ($I_0 = -0.005$, $\theta(0) = \theta_0^+ + 0.0001$), so that the neuron fires if not perturbed. For $w > 0$, the response curve is approximately linear, until it reaches zero. For $w < 0$, the current might cancel the spike if it occurs early.

Figure 2 shows how the firing time of a theta neuron changes with the time of arrival of a synaptic current. In our time coding model, we view this curve as the transfer function of the neuron; it describes how the neuron converts input spike times into output spike times.

## 2.3 Learning rule

We derive a spike-timing dependent learning rule from the objective of learning a set of target firing times. Following [2], we consider the mean squared error, $E$, between desired spike times $\bar{t}_s$ and actual spike times $t_s$:

$$E = < (t_s - \bar{t}_s)^2 > \tag{3}$$

where $< . >$ denotes the mean. Gradient descent on $E$ yields the following stochastic learning rule:

$$\Delta w_i = -\eta \frac{\partial E}{\partial w_i} = -2\eta(t_s - \bar{t}_s) \frac{\partial t_s}{\partial w_i} \tag{4}$$

The partial derivative $\frac{\partial t_s}{\partial w_i}$ expresses the credit assignment problem for synapses.

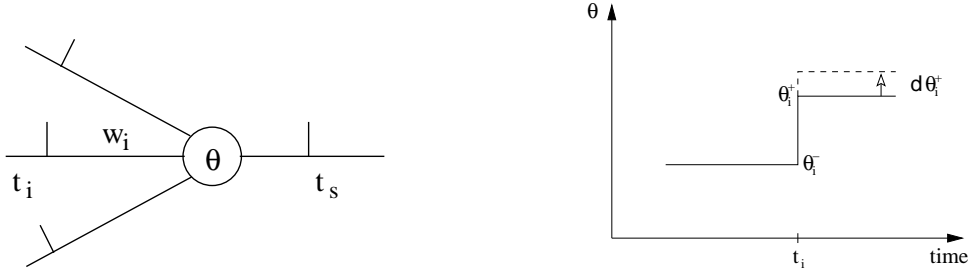

Figure 3: **Notations used in the text.** An incoming spike triggers an instantaneous change of the potential $\theta$. $\theta_i^-$ (resp. $\theta_i^+$) denotes the postsynaptic potential before (resp. after) the presynaptic spike. A small modification $dw_i$ of the synaptic weight $w_i$ induces a change $d\theta_i^+$

Let $F$ denote the "remaining time", that is, the time that remains before the neuron will fire:

$$F(t) = \int_{\theta(t)}^{\pi} \frac{d\theta}{(1 - cos\theta) + \alpha I(1 + cos\theta)} \tag{5}$$

In our model, $I$ is not continuous, because of Dirac synaptic currents. For the moment, we assume that $\theta$ is between the unstable point $\theta_0^+$ and $\pi$. In addition, we assume that the neuron receives one spike on each of its synapses, and that all synaptic weights are positive. Let $t_j$ denote the time of arrival of the action potential on synapse $j$. Let $\theta_j^-$ (resp. $\theta_j^+$) denote the potential before (resp. after) the synaptic current:

$$\begin{cases} \theta_j^- = \theta(t_j^-) \\ \theta_j^+ = \theta(t_j^+) = \theta_j^- + \alpha w_j(1 + \cos \theta_j^-) \end{cases} \tag{6}$$

We consider the effect of a small change of weight $w_i$. We shall rewrite integral (5) on the intervals where the integrand is continuous. To keep notations simple, we assume that action potentials are ordered, $ie : t_j \leq t_{j+1}$ for all $j$. For consistency, we use the notation $\theta_{N+1}^- = \pi$. We may write:

$$F(t_i) = \sum_{j \geq i} \int_{\theta_j^+}^{\theta_{j+1}^-} \frac{d\theta}{(1 - cos\theta) + \alpha I_0(1 + cos\theta)} \tag{7}$$

The partial derivative of the spiking time $t_s$ can be expressed as :

$$\frac{\partial t_s}{\partial w_i} = \frac{\partial F}{\partial \theta_i^+} \frac{\partial \theta_i^+}{\partial w_i} + \sum_{j > i} \left( \frac{\partial F}{\partial \theta_j^+} \frac{\partial \theta_j^+}{\partial w_i} + \frac{\partial F}{\partial \theta_j^-} \frac{\partial \theta_j^-}{\partial w_i} \right) \tag{8}$$

In this expression, the sum expresses how a change of weight $w_i$ will modify the effect of other spikes, for $j > i$. The $j^{th}$ terms of this sum depend on the time elapsed between $t_j$ and $t_i$. Since we have no *a priori* information on the distribution of $t_j$ given $t_i$, we shall consider that this term is not correlated with $\frac{\partial E}{\partial w_i}$. For that reason, we neglect this sum in our stochastic learning rule:

$$\frac{\partial t_s}{\partial w_i} \approx \frac{\partial F}{\partial \theta_i^+} \frac{\partial \theta_i^+}{\partial w_i} \tag{9}$$

which yields :

$$\frac{\partial t_s}{\partial w_i} \approx -\frac{(1 + \cos \theta_i^-)\alpha}{(1 - \cos \theta_i^+) + \alpha I_0(1 + \cos \theta_i^+)} \tag{10}$$

Note that this expression is not bounded when $\theta_i^+$ is close to the unstable point $\theta_0^+$. In that case, $\theta$ is in a region where it changes very slowly, and the timing of other action potentials for $j > i$ will mostly determine the firing time $t_s$. This means that approximation (9) will not hold. In addition, it is necessary to extend the learning rule to the case $\theta_i^+ \in [\theta_0^- \theta_0^+[$, where the above expression is negative. For these reasons, we introduce a *credit bound*, $C$, and we modify the learning rule as follows:

$$\text{if } 0 < -\frac{\partial t_s}{\partial w_i} < C \quad \text{then:} \quad \Delta w_i = -2\eta(t_s - \bar{t}_s)\frac{\partial t_s}{\partial w_i} \tag{11}$$

$$\text{else:} \quad \Delta w_i = 2\eta(t_s - \bar{t}_s)C \tag{12}$$

### 2.4 Algorithm

The algorithm updates the weights in the direction of the gradient. The learning rule takes effect at the end of a $trial$ of fixed duration. If a neuron does not fire at all during the trial, then its firing time is considered to be equal to the duration of the trial.

For each synapse, it is necessary to compute the credit from Equation (10) everytime a current is transmitted. We may relax the assumption that each synapse receives one single action potential; if a presynaptic neuron fires several times before the postsynaptic neuron fires, then the credit corresponding to all spikes is summed.

Theta neurons were simulated using Euler integration of Equation (1). The time step must be carefully chosen; if the temporal resolution is too coarse, then the credit assignment problem becomes too difficult, which increases the number of trials necessary for learning. On the other hand, small values of the time step mean that simulations take more time.

## 3 Auto-encoder network

Predicting neural activities has been proposed as a possible role for spike-timing dependent learning rules [7]. Here we train a network to predict its own activities using the learning rule derived above. For this, a time-delayed version of the input (echo) is used as the desired output (see Figure 4). The network has to find a representation of the input that minimizes mean squared reconstruction error.

The network has three populations of neurons: (i) An input population $X$ of size $n$ neurons, where an input vector is represented using spike times. We call Inter Stimulus Interval (ISI) the interval between the spikes encoding the input and the echo. After the ISI, population $X$ fires a second burst of spikes, that is a time-delayed version of the initial burst. (ii) An output population $Y$, of size $m$ neurons, that is activated by neurons in $X$. (iii) A population $X'$ of size $n$ neurons, where the input is reconstructed. Neurons in $X'$ are activated by $Y$. The learning rule updates the feedback connections $(w_{ij})_{i \leq n, j \leq m}$ from $Y$ to $X$, comparing spike times in $X$ and in $X'$.

We use $I_0 < 0$, so the response to positive transient currents is approximately linear (see fig. 2). We thus expect neurons to perform linear summation of spike times. For the feed-forward connections from $X$ to $Y$, we use the transpose of the feedback weights matrix. This is inspired by Oja's Principal Subspace Network [8]. If spike times are within the linear part of the response curve, then we expect this network to perform Principal Component Analysis (PCA) in the time domain.

However, one difference is that the PRC we use is always positive (type I neurons). This means that spike times can only code for positive values (even though synaptic weights can be of both signs).

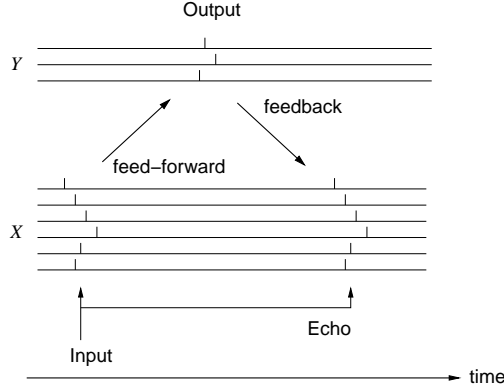

Figure 4: **Auto-encoder network.** An input vector is translated into firing times of the input population. Output neurons are activated by input neurons through feed-forward connections. A reconstruction of the input burst is generated through feedback connections. Target firing times are provided by a delayed version of the input burst (echo).

In order to code for values of both signs, one would need a transfer function that changes its sign around a time that would code for zero, so that the effect of a current is reversed when its arrival time crosses zero. Here we may view the neural code as a positive code: Early spikes code for high values, and late spikes code for values close to zero.

In this architecture, it is necessary to ensure that each neuron in $Y$ fires a single spike on each trial. In order to do this, we impose that neurons in $Y$ have the same average firing time. For this, we add a centering term to the learning rule:

$$\Delta w_{ij} = -\eta \frac{\partial E}{\partial w_{ij}} - \lambda \phi_j \tag{13}$$

where $\lambda \in \mathbb{R}$ and $\phi_j$ is the *average phase* of neuron $j$. $\phi_j$ is a leaky average of the difference between the firing time $t_j$ and the average firing times of all neurons in population $Y$. It is updated after each trial:

$$\phi_j \leftarrow \tau \phi_j + (1 - \tau) \left( t_j - \frac{1}{m} \sum_{k=1}^{m} t_k \right) \tag{14}$$

This modification of the learning rule results in neurons that have no preferred firing order.

## 4 Experiments

We used $I_0 = -0.01$ for all neurons. This ensures that neurons have no spontaneous activity. At the beginning of a trial, all neurons were initialized to their stable fixed point. In order to balance the effect of the different sizes of populations $X$ and $Y$, different values of $\alpha$ were used for $X$ and $Y$ neurons: We used $\alpha_X = 0.1$ and $\alpha_Y = \frac{m}{n} \alpha_X$. In the leaky average we used $\tau = 0.1$

In each experiment, the input vector was encoded in spike times. When doing so, one must make sure that the values taken by the input are within the coding interval of the neurons, *ie* the range of values where the PRC is not zero. In practice, spikes that arrive too late in the firing cycle are not taken into account by the learning rule. In that case, the weights corresponding to other synapses become overly increased, which eventually causes some postsynaptic neurons in $X'$ to fire *before* presynaptic neurons in $Y$ ("anticausal spikes"). If this occurs, one possibility is to reduce the variance of the input.

### 4.1 Principal Component Analysis of a Gaussian distribution

A two-dimensional Gaussian random variable was encoded in the spike times of three input neurons. The ellipsoid had a long axis of standard deviation $1ms$ and a short axis of deviation $0.5ms$, and it

was rotated by $\pi/3$. Because the network does not have an absolute time reference, it is necessary to use three input neurons, in order to encode two degrees of freedom in relative spiking times. The output layer had two neurons (one degree of freedom). Therefore the network has to find a 1D representation of a 2D variable, that minimizes the mean-squared reconstruction error.

The input was encoded as follows:

$$\begin{cases} t_0 = 3 \\ t_1 = 3 + \nu_1 \cos(\pi/3) + 0.5\nu_2 \sin(\pi/3) \\ t_2 = 3 + 0.5\nu_2 \cos(\pi/3) + \nu_1 \sin(\pi/3) \end{cases} \tag{15}$$

where $\nu_1$ and $\nu_2$ are two independent random variables picked from a Gaussian distribution of variance 1. Input spikes times were centered around $t = 3ms$, where $t = 0$ denotes the beginning of a trial. We used a time step of 0.05 ms. Each trial lasted for 400 iterations, which corresponds to $20ms$ of simulated time. The ISI was $5ms$. The credit bound was $C = 1000$. Other parameters were $\eta = 0.0001$ and $\lambda = 0.001$. Weights were initialized with random values between $0.5$ and $1.5$.

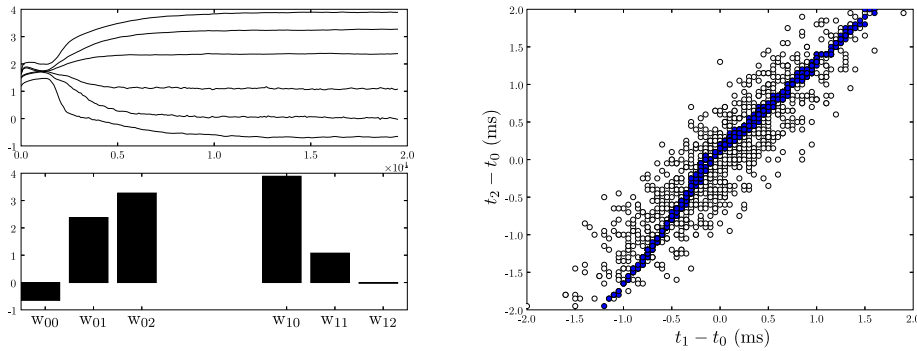

Figure 5: **Principal Component Analysis of a 2D Gaussian distribution.** The input vector was encoded in the relative spike times of three input neurons. Top: Evolution of the weights over 20.000 learning iterations. Bottom: Final synaptic weights represented as bars. Note the complementary shapes of weight vectors. Right: The input (white dots) and its reconstruction (dark dots) from the network's activities. Each branch corresponds to a firing order of the two output neurons.

Figure 5 shows that the network has learned to extract the principal direction of the distribution. Two branches are visible in the distribution of dots corresponding to the reconstruction. They correspond to two firing orders of the output neurons. The direction of the branches results from the synaptic weights of the neurons. Note that the lower branch has a slight curvature. This suggests that the response function of neurons is not perfectly linear in the interval where spike times are coded. The fact that branches do not exactly have the same orientation might result from non-linearities, or from the approximation made in deriving the learning rule.

There are six synaptic weights in the network. One degree of freedom per neuron in $X'$ is used to adapt its mean firing times to the value imposed by the ISI; the smaller the ISI, the larger the weights. This "normalization" removes three degrees of freedom. One additional constraint is imposed by the centering term that was added to the learning rule in (13). Thus the network had two degrees of freedom. It used them to find the directions of the two branches shown in Figure 5 (left). These two branches can be viewed as the base vectors used in the compressed representation in $Y$.

The network uses two base vectors in order to represent one single principal direction; each codes for one half of the Gaussian. This is because the network uses a positive code, where negative values are not allowed.

## 4.2  Encoding natural images

An encoder network was trained on the set of raw natural images used in [9][1]. The encoder had 64 output neurons and 256 input neurons. On each trial, a random patch of size $16 \times 16$ was extracted

from a random image of the dataset, and encoded in the network. Raw grey values from the dataset were encoded as milliseconds. The standard deviation per pixel was $1.00ms$. The time step of the simulation was $0.1ms$, and each trial lasted for 200 time steps ($20ms$). The ISI was $9ms$, and the parameters of the learning rule were $\eta = 0.0001$, $C = 50$ and $\lambda = 0.001$. Weights were initialized with random values between 0 and 0.3.

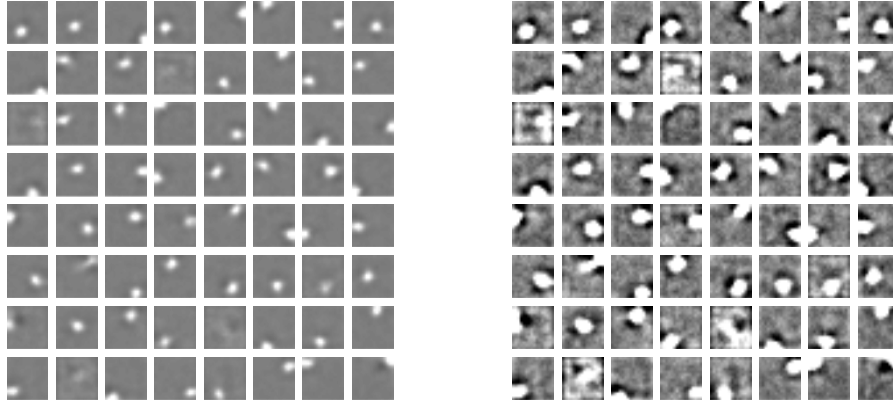

Figure 6: **Synaptic weights learned by the network**. 64 neurons were trained to represent natural images patches of size $16 \times 16$. Different grey scales are used in order to display positive and negative weights (black is negative, white is positive). Left: grey scale between -1 and 1. Only positive weights are visible at this scale, because they are much larger than negative weights. Right: grey scale between -0.1 and 0.1. Negative weights are visible, positive weights are beyond scale.

Synaptic weights after 100.000 trials are shown in Figure 6. There is a strong difference of amplitude between positive and negative weights; positive weights typically have values between 0 and 1, while negative weights are one order of magnitude smaller. For that reason, weights are displayed twice, with two different grey scales. An image reconstructed from spike times is shown in Figure 7. After training, the mean reconstruction error on the entire dataset was $0.25ms$/pixel. For comparison, the mean error performed by Oja's principal subspace network [8] trained on the same image patches was $0.11ms$/pixel.

The difference of amplitude between positive and negative weights results from higher sensitivity of the response curves to negative weights, as shown in Figure 2. Synaptic weights with negative values have the ability to strongly delay the output spike, and even to cancel it.

Synaptic weights have the shape of local filters, with antagonistic center-surround structures. This contrasts with the base vectors typically obtained from PCA of natural images, which are not local. One possible explanation lies in the response properties of the theta neurons. The response function is not linear, especially in the case of negative weights (Figure 2). This will disfavor solutions involving linear combinations of both positive and negative weights, and favor sparse representations. Hence, the network could be performing something similar to Nonlinear PCA [10].

# 5   Conclusions

We have shown that the dynamic response properties of spiking neurons can be effectively used as transfer functions, in order to perform computations (in this paper, PCA and Nonlinear PCA). A similar proposal was made in [11], where the PRC of neurons has been adapted to a biologically realistic STDP rule. Here we took a complementary approach, adapting the learning rule to the neuronal dynamics.

We used theta neurons, which are of type I, and equivalent to quadratic integrate-and-fire neurons. Type I neurons have a PRC that is always positive. This means that spike times can encode only

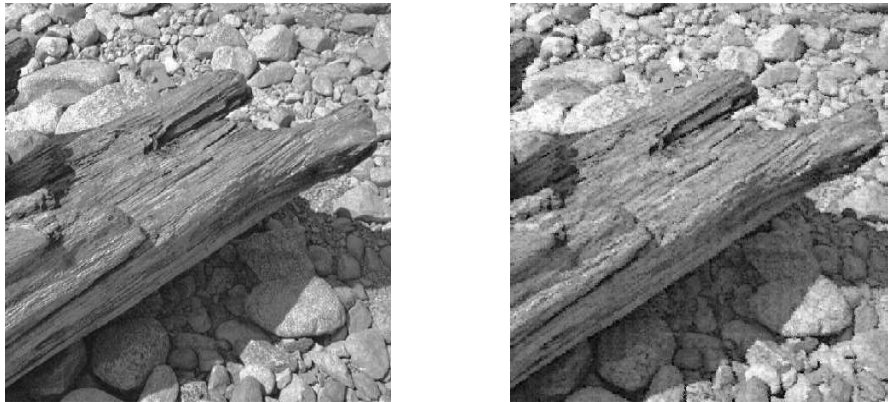

Figure 7: **Natural image and reconstruction from spike times**. The $512 \times 512$ image from the training set (left) was divided into $16 \times 16$ patches, and encoded using 64 neurons. The reconstruction (right) is derived from spikes times in $X'$. Standard deviation of the encoded images was $1.00 ms$/pixel. The mean reconstruction error on the entire dataset was $0.25 ms$/pixel, about 2.5 times the error made by PCA.

positive values. In order to encode values of both signs, one would need the transfer function to change its sign around a time that codes for zero. This will be possible with more complex type II neurons, where the sign of the PRC is not constant.

### Acknowledgments

The author thanks Samuel McKennoch and Dominique Martinez for helpful comments.

## Footnotes

[1]Images were retrieved from http://redwood.berkeley.edu/bruno/sparsenet/

### References

[1] W. Maass. Lower bounds for the computational power of networks of spiking neurons. *Neural Computation*, 8(1):1–40, 1996.

[2] S.M. Bohte, J.N. Kok, and H. La Poutré. Spike-prop: error-backprogation in multi-layer networks of spiking neurons. *Neurocomputing*, 48:17–37, 2002.

[3] A. J. Bell and L. C. Parra. Maximising sensitivity in a spiking network. In *Advances in Neural Information Processing Systems*, volume 17, pages 121–128, 2005.

[4] R. F. Galán, G. B. Ermentrout, and N. N. Urban. Efficient estimation of phase-resetting curves in real neurons and its significance for neural-network modeling. *Physical Review Letters*, 94:158101, 2005.

[5] G. B. Ermentrout. Type I membranes, phase resetting curves, and synchrony. *Neural Computation*, 8:979–1001, 1996.

[6] W. Gerstner and W. M. Kistler. *Spiking Neuron Models : Single Neurons, Populations, Plasticity.* Cambridge University Press, 2002.

[7] R. P. N. Rao and T. J. Sejnowski. Predictive sequence learning in recurrent neocortical circuits. In *Advances in Neural Information Processing Systems*, volume 12, pages 164–170, 2000.

[8] E. Oja. Neural networks, principal components and subspaces. *International Journal of Neural Systems*, 1(1):61–68, 1989.

[9] B. Olshausen and D. Field. Sparse coding of natural images produces localized, oriented, bandpass receptive fields. *Nature*, 381:607–609, 1996.

[10] E. Oja. The nonlinear PCA learning rule in independent component analysis. *Neurocomputing*, 17(1):25–46, 1997.

[11] Lengyel M., Kwag J., Paulsen O., and Dayan P. Matching storage and recall:hippocampal spike timing-dependent plasticity and phase response curves. *Nature Neuroscience*, 8:1677–1683, 2006.
